# Spatial and anatomical regularization of SVM for brain image analysis

**Rémi Cuingnet**
CRICM (UPMC/Inserm/CNRS), Paris, France
Inserm - LIF (UMR_S 678), Paris, France
`remi.cuingnet@imed.jussieu.fr`

**Marie Chupin**
CRICM, Paris, France
`marie.chupin@upmc.fr`

**Habib Benali**
Inserm - LIF, Paris, France
`habib.benali@imed.jussieu.fr`

**Olivier Colliot**
CRICM, Paris, France
`olivier.colliot@upmc.fr`

## Abstract

Support vector machines (SVM) are increasingly used in brain image analyses since they allow capturing complex multivariate relationships in the data. Moreover, when the kernel is linear, SVMs can be used to localize spatial patterns of discrimination between two groups of subjects. However, the features' spatial distribution is not taken into account. As a consequence, the optimal margin hyperplane is often scattered and lacks spatial coherence, making its anatomical interpretation difficult. This paper introduces a framework to spatially regularize SVM for brain image analysis. We show that Laplacian regularization provides a flexible framework to integrate various types of constraints and can be applied to both cortical surfaces and 3D brain images. The proposed framework is applied to the classification of MR images based on gray matter concentration maps and cortical thickness measures from 30 patients with Alzheimer's disease and 30 elderly controls. The results demonstrate that the proposed method enables natural spatial and anatomical regularization of the classifier.

## 1 Introduction

Brain image analyses have widely relied on univariate voxel-wise analyses, such as voxel-based morphometry (VBM) for structural MRI [1]. In such analyses, brain images are first spatially registered to a common stereotaxic space, and then mass univariate statistical tests are performed in each voxel to detect significant group differences. However, the sensitivity of theses approaches is limited when the differences are spatially complex and involve a combination of different voxels or brain structures [2]. Recently, there has been a growing interest in support vector machines (SVM) methods [3, 4] to overcome the limits of these univariate analyses. Theses approaches allow capturing complex multivariate relationships in the data and have been successfully applied to the individual classification of a variety of neurological conditions [5, 6, 7, 8]. Moreover, the output of the SVM can also be analyzed to localize spatial patterns of discrimination, for example by drawing the coefficients of the optimal margin hyperplane (OMH) – which, in the case of a linear SVM, live in the same space as the MRI data [7, 8]. However, one of the problems with analyzing directly the OMH coefficients is that the corresponding maps are scattered and lack spatial coherence. This makes it difficult to give a meaningful interpretation of the maps, for example to localize the brain regions altered by a given pathology.

In this paper, we address this issue by proposing a framework to introduce spatial consistency into SVMs by using regularization operators. Section 2 provides some background information on SVMs

and regularization operators. We then show that the regularization operator framework provides a flexible approach to model different types of proximity (section 3). Section 4 presents the first type of regularization, which models spatial proximity, i.e. two features are close if they are spatially close. We then present in section 5 a more complex type of constraint, called anatomical proximity. In the latter case, two features are considered close if they belong to the same brain network; for instance two voxels are close if they belong to the same anatomical or functional region or if they are anatomically or functionally connected (based on fMRI networks or white matter tracts). Finally, in section 6, the proposed framework is illustrated on the analysis of MR images using gray matter concentration maps and cortical thickness measures from 30 patients with AD and 30 elderly controls from the ADNI database (www.adni-info.org).

## 2 Priors in SVM

In this section, we first describe the neuroimaging data that we consider in this paper. Then, after some background on SVMs and on how to add prior knowledge in SVMs, we describe the framework of regularization operators.

### 2.1 Brain imaging data

In this contribution, we consider any feature computed either at each voxel of a 3D brain image or at any vertex of the cortical surface. Typically, for anatomical studies, the features could be tissue concentration maps such as gray matter (GM) or white matter (WM) for the 3D case or cortical thickness maps for the surface case. The proposed methods are also applicable to functional or diffusion weighted MRI. We further assume that 3D images or cortical surfaces were spatially normalized to a common stereotaxic space (e.g. [9]) as in many group studies or classification methods [5, 6, 7, 8, 10].

Let $\mathcal{V}$ be the domain of the 3D images or surfaces. $v$ will denote an element of $\mathcal{V}$ (i.e. a voxel or a vertex). Thus, $\mathcal{X} = \mathbb{R}^{\mathcal{V}}$, together with the canonical dot product will be the *input space*.

Let $\mathbf{x}_s \in \mathcal{X}$ be the data of a given subject $s$. In the case of 3D images, $\mathbf{x}_s$ can be considered in two different ways: $(i)$ as an element of $\mathbb{R}^d$ where $d$ denotes the number of voxels, $(ii)$ as a real-valued function defined on a compact subset of $\mathbb{R}^3$. Both finite and continuous viewpoints will be studied in this paper because they allow different types of regularization. Similarly, in the surface case, $\mathbf{x}_s$ can be viewed either as an element of $\mathbb{R}^d$ where $d$ denotes the number of vertices or as a real-valued function on a 2-dimensional compact Riemannian manifold.

We consider a group of $N$ subjects with their corresponding data $(\mathbf{x}_s)_{s \in [1,N]} \in \mathcal{X}^N$. Each subject is associated with a group $(y_s)_{s \in [1,N]} \in \{-1, 1\}^N$ (typically his diagnosis, i.e. diseased or healthy).

### 2.2 Linear SVM

The linear SVM solves the following optimization problem [3, 4, 11]:

$$\left( \mathbf{w}^{\mathrm{opt}}, b^{\mathrm{opt}} \right) = \underset{\mathbf{w} \in \mathcal{X}, b \in \mathbb{R}}{\arg \min} \sum_{s=1}^{N} l_{\mathrm{hinge}} \left( y_s \left[ \langle \mathbf{w}, \mathbf{x}_s \rangle + b \right] \right) + \lambda \parallel \mathbf{w} \parallel^2 \tag{1}$$

where $\lambda \in \mathbb{R}^+$ is the *regularization parameter* and $l_{\mathrm{hinge}}$ the *hinge loss function* defined as: $l_{\mathrm{hinge}} : u \in \mathbb{R} \mapsto \max(0, 1 - u)$.

With a linear SVM, the *feature space* is the same as the *input space*. Thus, when the input features are the voxels of a 3D image, each element of $\mathbf{w}^{\mathrm{opt}} = (w_v^{\mathrm{opt}})_{v \in \mathcal{V}}$ also corresponds to a voxel. Similarly, for the surface-based methods, the elements of $\mathbf{w}^{\mathrm{opt}}$ can be represented on the vertices of the cortical surface. To be anatomically consistent, if $v^{(1)} \in \mathcal{V}$ and $v^{(2)} \in \mathcal{V}$ are close according to the topology of $\mathcal{V}$, their weights in the SVM classifier, $w_{v^{(1)}}^{\mathrm{opt}}$ and $w_{v^{(2)}}^{\mathrm{opt}}$ respectively, should be similar. In other words, if $v^{(1)}$ and $v^{(2)}$ correpond to two neighboring regions, they should have a similar role in the classifier function. However, this is not guaranteed with the standard linear SVM (as for example in [7]) because the regularization term *is not a spatial regularization*. The aim of the present paper is to propose methods to ensure that $\mathbf{w}^{\mathrm{opt}}$ is spatially regularized.

## 2.3 How to include priors in SVM

To spatially regularize the SVM, one has to include some prior knowledge on the proximity of features. In the literature, three main ways have been considered in order to include priors in SVMs.

In an SVM, all the information used for classification is encoded in the kernel. Hence, the first way to include prior is to directly design the kernel function [4]. But this implies knowing a metric on the input space $\mathcal{X}$ consistent with the prior knowledge.

Another way is to force the classifier function to be locally invariant to some transformations. This can be done: $(i)$ by directly engineering a kernel which leads to locally invariant SVM, $(ii)$ by generating artificially transformed examples from the training set to create virtual support vectors (virtual SV), $(iii)$ by using a combination of both these approaches called kernel jittering [12, 13, 14]. But the main difficulty here is how to define the transformations to which we would like the kernel to be invariant.

The last way is to consider SVM from the regularization viewpoint [15, 4]. The idea is to force the classifier function to be smooth with respect to some criteria. This is the viewpoint which is adopted in this paper.

## 2.4 Regularization operators

Our aim is to introduce a spatial regularization on the classifier function of the SVM which can be written as $\mathrm{sgn}\,(f(\mathbf{x}_s) + b)$ where $f \in \mathbb{R}^{\mathcal{X}}$. This is done through the definition of a *regularization operator* $P$ on $f$. Following [15, 4], $P$ is defined as a linear map from a space $\mathcal{F} \subset \mathbb{R}^{\mathcal{X}}$ into a dot product space $(\mathcal{D}, \langle \cdot, \cdot \rangle_{\mathcal{D}})$.

$G : \mathcal{X} \times \mathcal{X} \to \mathbb{R}$ is a Green's function of a regularization operator P iff:

$$\forall f \in \mathcal{F},\ \forall \mathbf{x} \in \mathcal{X},\ f(\mathbf{x}) = \langle P(G(\mathbf{x}, \cdot)), P(f) \rangle_{\mathcal{D}} \tag{2}$$

If $P$ admits at least a Green's function called $G$, then $G$ is a positive semi-definite kernel and the minimization problem:

$$\left( f^{\mathrm{opt}}, b^{\mathrm{opt}} \right) = \underset{f \in \mathcal{F}, b \in \mathbb{R}}{\arg\min} \sum_{s=1}^{N} l_{\mathrm{hinge}} \left( y_s \left[ f(\mathbf{x}_s) + b \right] \right) + \lambda \parallel P(f) \parallel_{\mathcal{D}}^2 \tag{3}$$

is equivalent to the SVM minimization problem with kernel $G$.

Since in linear SVM, the feature space is the input space, $f$ lies in the input space. Therefore, the optimisation problem (3) is very convenient to include spatial regularization on $f$ via the definition of $P$. Note that, usually, $\mathcal{F}$ is a Reproducing Kernel Hilbert Space (RKHS) with kernel $K$ and $\mathcal{D} = \mathcal{F}$. Hence, if $P$ is bounded, injective and compact, $P$ admits a Green's function $G = (P^{\dagger}P)^{-1}K$ where $P^{\dagger}$ denotes the adjoint of $P$.

One has to define the regularization operator $P$ so as to obtain the suitable regularization for the problem.

# 3 Laplacian regularization

Spatial regularization requires the notion of proximity between elements of $\mathcal{V}$. This can be done through the definition of a graph in the discrete case or a metric in the continuous case. In this section, we propose spatial regularizations based on the Laplacian for both of these proximity models. This penalizes the high-frequency components with respect to the topology of $\mathcal{V}$.

## 3.1 Graphs

When $\mathcal{V}$ is finite, weighted graphs are a natural framework to take spatial information into consideration. Voxels of a brain image can be considered as nodes of a graph which models the voxels' proximity. This graph can be the voxel connectivity (6, 18 or 26) or a more sophisticated graph.

We chose the following regularization operator:

$$P : \ \mathbf{w}^* \in \mathcal{F} = \mathcal{L}(\mathbb{R}^{\mathcal{V}}, \mathbb{R}) \mapsto \left( e^{\frac{1}{2}\beta L} \mathbf{w} \right)^* \in \mathcal{F} \tag{4}$$

where $L$ denotes the graph Laplacian [16] and $\mathbf{w}^*$ the dual vector of $\mathbf{w}$. $\beta$ controls the size of the regularization. The optimization problem then becomes:

$$(\mathbf{w}^{\text{opt}}, b^{\text{opt}}) = \underset{\mathbf{w} \in \mathcal{X}, b \in \mathbb{R}}{\arg \min} \sum_{s=1}^{N} l_{\text{hinge}} \left( y_s \left[ \langle \mathbf{w}, \mathbf{x}_s \rangle + b \right] \right) + \lambda \parallel e^{\frac{1}{2}\beta L} \mathbf{w} \parallel^2 \qquad (5)$$

Such a regularization exponentially penalizes the high-frequency components and thus forces the classifier to consider as similar voxels highly connected according to the graph adjacency matrix. According to the previous section, this new minimization problem (5) is equivalent to an SVM optimization problem. The new kernel $K_\beta$ is given by:

$$K_\beta(\mathbf{x}_1, \mathbf{x}_2) = \mathbf{x}_1^T e^{-\beta L} \mathbf{x}_2 \qquad (6)$$

This is a heat or diffusion kernel on a graph. Our approach differs from the diffusion kernels introduced by Kondor et al. [17] because the nodes of the graph are the features, here the voxels, whereas in [17], the nodes were the objects to classify. Laplacian regularization was also used in satellite imaging [18] but, again, the nodes were the objects to classify. Our approach can also be considered as a spectral regularization on the graph [19]. To our knowledge, such spectral regularization has not been applied to brain images but only to the classification of microarray data [20].

## 3.2 Compact Riemannian manifolds

In this paper, when $\mathcal{V}$ is continuous, it can be considered as a 2-dimensional (e.g. surfaces) or a 3-dimensional (e.g. 3D Euclidean or more complex) compact Riemannian manifold. The metric then models the notion of proximity. On such spaces, the heat kernel exists [21, 22]. Therefore, the Laplacian regularization presented in the previous paragraph can be extended to compact Riemannian manifolds [22]. Similarly to the graphs, we chose the following regularization operator:

$$P : \mathbf{w}^* \in \mathcal{F} = \mathcal{L}(\mathbb{R}^{\mathcal{V}}, \mathbb{R}) \mapsto \left( e^{\frac{1}{2}\beta \Delta} \mathbf{w} \right)^* \in \mathcal{F} \qquad (7)$$

where $\Delta$ denotes the Laplace-Beltramin operator. The optimization problem is also equivalent to an SVM optimization problem with kernel $K_\beta(\mathbf{x}_1, \mathbf{x}_2) = \mathbf{x}_1^T e^{-\beta \Delta} \mathbf{x}_2$. Note the difference between our approach and that of Laferty and Lebanon [22]. In our case, the points of the manifolds are the features, whereas in [22], they were the objects to classify.

In sections 4 and 5, we present different types of proximity models which correspond to different types of graphs or distances.

# 4 Spatial proximity

In this section, we consider the case of regularization based on spatial proximity, i.e. two voxels (or vertices) are close if they are spatially close.

## 4.1 The 3D case

When $\mathcal{V}$ are the image voxels (discrete case), the simplest option to encode the spatial proximity is to use the image connectivity (e.g. 6-connectivity) as a regularization graph. Similarly, when $\mathcal{V}$ is a compact subset of $\mathbb{R}^3$ (continuous case), the proximity is encoded by a Euclidean distance. In both cases, this is equivalent to pre-process the data with a Gaussian smoothing kernel with standard deviation $\sigma = \sqrt{\beta}$ [17].

However, smoothing the data with a Gaussian kernel would mix gray matter (GM), white matter (WM) and cerebrospinal fluid (CSF). Instead, we propose a graph which takes into consideration both the spatial localization and the tissue types. Based on tissue probability maps, in each voxel $v$, we have the set of probabilities $p_v$ that this voxel belongs to GM, WM or CSF. We considered the following graph. Two voxels are connected if and only if they are neighbors in the image (6-connectivity). The weight $a_{u,v}$ of the edge between two connected voxels $u$ and $v$ is $a_{u,v} = e^{-d_{\chi^2}(p_u, p_v)^2 / (2\sigma^2)}$, where $d_{\chi^2}$ is the $\chi^2$-distance between two distributions. We chose beforehand $\sigma$ equal to the standard deviation of $d_{\chi^2}(p_u, p_v)$.

To compute the kernel, we computed $e^{-\beta L} \mathbf{x}_s$ for each subject $s$ in the training set by scaling the Laplacian and using the Taylor series expansion.

## 4.2 The surface case

The connectivity graph is not directly applicable to surfaces. Indeed, the regularization would then strongly depend on the mesh used to discretize the surface. This shortcoming can be overcome by reweighing the graph with conformal weights. In this paper, we chose a different approach by adopting the continuous viewpoint: we consider the cortical surface as a 2-dimensional Riemannian manifold and use the regularization operator defined by equation (7). Indeed, the Laplacian is an intrinsic operator and does not depend on the chosen surface parameterization. The heat kernel has already been used for cortical smoothing for example in [23, 24, 25, 26]. We will therefore not detail this part. We used the implementation described in [26].

# 5 Anatomical proximity

In this section, we consider a different type of proximity, which we call anatomical proximity. Two voxels are considered close if they belong to the same brain network. For example, two voxels can be close if they belong to the same anatomical or functional region (defined for example by a probabilistic atlas). This can be seen as a "short-range" connectivity. Another example is that of "long-range" proximity which models the fact that distant voxels can be anatomically (through white matter tracts) or functionally connected (based on fMRI networks).

We first focus on the discrete case. The presented framework can be used either for 3D images or surfaces and computed very efficiently. However, such an efficient implementation was obtained at the cost of the spatial proximity. Therefore, we then show a continuous formulation which enables to consider both spatial and anatomical proximity.

## 5.1 On graphs: atlas and connectivity

Let $(\mathcal{A}_1, \cdots, \mathcal{A}_R)$ be the $R$ regions of interest (ROI) of an atlas and $p(v \in A_r)$ the probability that the voxel $v$ belongs to region $\mathcal{A}_r$. Then the probability that two voxels $v^{(i)}$ and $v^{(j)}$ belong to the same region is: $\sum_{r=1}^{R} p\left((v^{(i)}, v^{(j)}) \in \mathcal{A}_r^2\right)$. We assume that if $v^{(i)} \neq v^{(j)}$ then: $p\left((v^{(i)}, v^{(j)}) \in \mathcal{A}_r^2\right) = p\left(v^{(i)} \in \mathcal{A}_r\right) p\left(v^{(j)} \in \mathcal{A}_r\right)$. Let $E \in \mathbb{R}^{d \times R}$ be the right stochastic matrix defined by: $E_{i,r} = p\left(v^{(i)} \in \mathcal{A}_r\right)$. Then, for $i \neq j$, the $(i,j)$-th entry of the adjacency matrix $EE^t$ is the probability that the voxels $v^{(i)}$ and $v^{(j)}$ belong to the same regions.

For "long-range" connections (structural or functional), one can consider an $R$-by-$R$ matrix $C$ with the $(r_1, r_2)$-th entry being the probability that $\mathcal{A}_{r_1}$ and $\mathcal{A}_{r_2}$ are connected. Then the adjacency matrix becomes: $ECE^t$. We considered the normalized Laplacian $\tilde{L}$ [16], to be sure that the two terms commute:

$$\tilde{L} = I_d - D^{-\frac{1}{2}} ECE^t D^{-\frac{1}{2}} \tag{8}$$

where $D$ is a diagonal matrix. Hence, if $CE^t D^{-1} E$ is not singular, we have:

$$e^{-\beta \tilde{L}} = e^{-\beta} \left[ I_d + D^{-\frac{1}{2}} E(e^{\beta CE^t D^{-1}E} - I_R)(CE^t D^{-1}E)^{-1} CE^t D^{-\frac{1}{2}} \right] \tag{9}$$

The computation requires only the computation of $D^{-\frac{1}{2}}$, which is done efficiently since $D$ is a diagonal matrix, and the computation of inverse and the matrix exponential of an $R$-by-$R$ matrix, which is also efficient since $R \sim 10^2$.

This method can be directly applied to both 3D images and cortical surfaces. Unfortunately, the efficient implementation was obtained at the cost of the spatial proximity. The next section presents a combination of anatomical and spatial proximity using the continuous viewpoint.

## 5.2 On statistical manifolds

In this section, the goal is to take into account various prior informations such as tissue information, atlas information and spatial proximity. We first show that this can be done by considering the images or surfaces as statistical manifolds together with the Fisher metric. We then give some details about the computation of the kernel.

**Fisher metric** We assume that we are given an anatomical or a functional atlas $\mathcal{A}$ composed of $R$ regions: $\{\mathcal{A}_r\}_{r=1\cdots R}$. Similarly, $\mathcal{T} = \{\mathcal{T}_{\text{GM}}, \mathcal{T}_{\text{WM}}, \mathcal{T}_{\text{CSF}}\}$ denotes the set of brain tissues. In each point $v \in \mathcal{V}$, we have a probability distribution $p_{\text{atlas}}(\cdot|v) \in \mathbb{R}^{\mathcal{T} \times \mathcal{A}}$ which informs about the tissue type and the atlas region in $v$. Without any loss of generality, one can assume that the tissue information is encoded in the atlas. Therefore, we consider the probability $p_{\text{atlas}}(\cdot|v) \in \mathbb{R}^{\mathcal{A}}$. We also consider a probability distribution $p_{\text{loc}}(\cdot|v) \in \mathbb{R}^{\mathcal{V}}$ which encodes the spatial proximity. A simple example is $p_{\text{loc}}(\cdot|v) \sim \mathcal{N}(v, \sigma_{\text{loc}}^2)$. Therefore, we consider the probability family: $\mathcal{M} = \{p(\cdot|v) \in \mathbb{R}^{\mathcal{A} \times \mathcal{V}}\}_{v \in \mathcal{V}}$ where $p(\cdot|v) = p_{\text{atlas}}(\cdot|v)p_{\text{loc}}(\cdot|v)$.

A natural way to encode proximity on $\mathcal{M}$ is to use the Fisher metric as in [22]. With some smoothness assumption about $p$, $\mathcal{M}$ together with this metric is a compact Riemannian manifold [27]. For clarity, we present this framework only for 3D images but it could be applied to cortical surfaces with minor changes. The metric tensor $g$ is then given for all $v \in \mathcal{V}$ by:

$$g_{ij}(v) = \mathbb{E}_v \left[ \frac{\partial \log p(\cdot|v)}{\partial v_i} \frac{\partial \log p(\cdot|v)}{\partial v_j} \right], \ 1 \le i, j \le 3 \tag{10}$$

If we further assume that $p_{\text{loc}}(\cdot|v)$ is isotropic we have:

$$g_{ij}(v) = g_{ij}^{\text{atlas}}(v) + \delta_{ij} \int_{u \in \mathcal{V}} p_{\text{loc}}(u|v) \left( \frac{\partial \log p_{\text{loc}}(u|v)}{\partial v_i} \right)^2 du \tag{11}$$

where $\delta_{ij}$ is the Kronecker delta and $g^{\text{atlas}}$ is the metric tensor when $p(\cdot|v) = p_{\text{atlas}}(\cdot|v)$. When $p_{\text{loc}}(\cdot|v) \sim \mathcal{N}(v, \sigma_{\text{loc}}^2 I_3)$, we have: $g_{ij}(v) = g_{ij}^{\text{atlas}}(v) + \frac{\delta_{ij}}{\sigma_{\text{loc}}^2}$.

**Computing the kernel** Once the notion of proximity is defined, one has to compute the kernel matrix. The computation of the kernel matrix requires the computation of $e^{-\beta\Delta}\mathbf{x}_s$ for all the subjects of the training set. The eigendecomposition of the Laplace-Beltrami operator is intractable since the number of voxels in a brain images is about $10^6$. Hence $e^{-\beta\Delta}\mathbf{x}_s$ is considered as the solution at $t = \beta$ of the heat equation with the Dirichlet homogeneous boundary conditions:

$$\begin{cases} \frac{\partial \mathbf{u}}{\partial t} - \Delta \mathbf{u} = 0 \\ \mathbf{u}(t = 0) = \mathbf{x}_s \end{cases} \tag{12}$$

The Laplace-Beltrami operator is given by [21]: $\Delta \mathbf{u} = \frac{1}{\sqrt{\det g}} \sum_{j=1}^{3} \frac{\partial}{\partial v_j} \left( \sum_{i=1}^{3} h_{ij} \sqrt{\det g} \frac{\partial \mathbf{u}}{\partial v_i} \right)$

where $h$ is the inverse tensor of $g$.

To solve equation (12), one can use a variational approach [28]. We used the rectangular finite elements in space and the explicit finite difference scheme for the time discretization. $\Delta_x$ and $\Delta_t$ denote the space step and the time step respectively. $\Delta_x$ is fixed by the MRI spatial resolution. $\Delta_t$ is then chosen so as to respect the Courant-Friedrichs-Lewy (CFL) condition, which can be written in this case as: $\Delta_t \le 2(\max \lambda_i)^{-1}$, where $\lambda_i$ are the eigenvalues of the general eigenproblem: $\mathbf{K}U = \lambda\mathbf{M}U$ with $\mathbf{K}$ the stiffness matrix and $\mathbf{M}$ the mass matrix. To compute the optimal time step $\Delta_t$, we estimated the largest eigenvalue with the power iteration method.

## 6 Experiments and results

### 6.1 Material

**Subjects and MRI acquisition** Data were obtained from the Alzheimer's Disease Neuroimaging Initiative (ADNI) database [1]. The Principal Investigator of this initiative is Michael W. Weiner, M.D., VA Medical Center and University of California - San Francisco.For up-to-date information see www.adni-info.org. We studied 30 patients with probable AD (age$\pm$ standard-deviation (SD) = $74\pm4$, range = 60-80 years, mini-mental score (MMS) = $23\pm2$) and 30 elderly controls (age$\pm$ SD = $73\pm4$, range = 60-80, MMS = $29\pm1$) which were selected from the ADNI database according to the

following criteria. Subjects were excluded if their scan revealed major artifacts or gross structural abnormalities of the white matter, for it makes the tissue segmentation step fail. 80-year-old subjects or older were also excluded. The MR scans are T1-weighted MR images. MRI acquisition was done according to the ADNI acquisition protocol in [29].

**Features extraction** For the 3D image analyses, all T1-weighted MR images were segmented into gray matter (GM), white matter (WM) and cerebrospinal fluid (CSF) using the SPM5 (Statistical Parametric Mapping, London, UK) unified segmentation routine [30] and spatially normalized with DARTEL [9]. The features are the GM probability maps in the MNI space. For the surface-based analyses, the features are the cortical thickness values at each vertex of the cortical surface. Cortical thickness measures were performed with Freesurfer (Massachusetts General Hospital, Boston, MA).

## 6.2 Proposed experiments

As an illustration of the method, we present the results of the AD versus controls analysis. We present the maps associated to the optimal margin hyperplane (OMH). The classification function obtained with a linear SVM is the sign of the inner product of the features with $\mathbf{w}^{\mathrm{opt}}$, a vector orthogonal to the OMH [3, 4]. Therefore, if the absolute value of the $i^{\mathrm{th}}$ component of $\mathbf{w}^{\mathrm{opt}}$, $|w_i^{\mathrm{opt}}|$, is small compared to the other components $(|w_j^{\mathrm{opt}}|)_{j \neq i}$, the $i^{\mathrm{th}}$ feature will have a small influence on the classification. Conversely, if $|w_i^{\mathrm{opt}}|$ is relatively large, the $i^{\mathrm{th}}$ feature will play an important role in the classifier. Thus the optimal weights $\mathbf{w}^{\mathrm{opt}}$ allow us to evaluate the anatomical consistency of the classifier. In all experiments, the $C$ parameter of the SVM was fixed to one ($\lambda = \frac{1}{2NC}$ [4]).

## 6.3 Results: spatial proximity

In this section, we present the results for the spatial proximity in the 3D case (method presented in section 4.1). Due to space limitations, the surface case is not presented. Fig. 1(a) presents the OMH when no spatial regularization is performed. Fig. 1(b) shows the results with spatial proximity but without tissue probability maps. $\mathbf{w}$ becomes smoother and spatially consistent. However it mixes tissues and does not respect the topology of the cortex. For instance, it mixes tissues of the temporal lobe with tissues of the frontal and parietal lobes. The results with both spatial proximity and tissue maps are shown on Fig. 1(c). The OMH is much more consistent with the brain anatomy. $\beta$ controls the size of the spatial regularization and was chosen to be equivalent to a 4mm-FWHM of the Gaussian smoothing. The classification accuracy was estimated by a leave-one-out cross validation. The classifiers were able to distinguish AD from CN with similar accuracies (83% with no spatial priors and 85% with spatial priors).

## 6.4 Results: anatomical proximity

In this section, we present the results for the anatomical proximity. We first present the discrete surface case. The discrete 3D case leads to comparable results but is omitted here due to space limitations. We then present the continuous 3D case. Extension to surfaces is left for future work.

**Discrete case** For the discrete case, we used "short-range" proximity, defined by the cortical atlas of Desikan et al. [31] with binary probabilities. We tested different values for $\beta = 0, 1, \cdots, 5$. The accuracies ranged between 80% and 85%. The highest accuracy was reached for $\beta = 3$. The optimal SVM weights $\mathbf{w}$ are shown on Fig. 2. When no regularization has been carried out, they are noisy and scattered (Fig. 2 (a)). When the amount of regularization is increased, voxels of a same region tend to be considered as similar by the classifier (Fig. 2(b-d)). Note how the anatomical coherence of the OMH varies with $\beta$.

**Continuous case** We then present the results of the 3D continuous case (section 5.2). The atlas information used was only the tissue types. We chose $\sigma_{\mathrm{loc}} = 10$mm for the spatial confidency. $\beta$ was chosen to be equivalent to a 4mm-FWHM of the Gaussian smoothing. The classifier reached 87% accuracy. The optimal SVM weights $w$ are shown on Fig. 1(d). The tissue knowledge enables the classifier to be more consistent with the anatomy. For instance, note the difference with the Gaussian smoothing (Fig. 1(b)) and how the proposed method avoids mixing the temporal lobe with the parietal and frontal lobes.

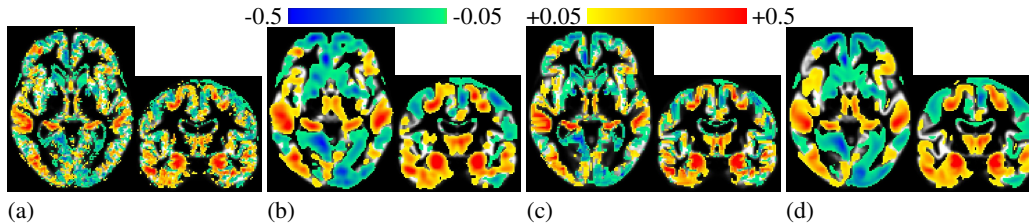

Figure 1: Normalized **w** coefficients: (a) no spatial prior, (b) spatial proximity: FWHM=4mm, (c) spatial proximity and tissues: FWHM~4mm, (d) Fisher metric using tissue maps.

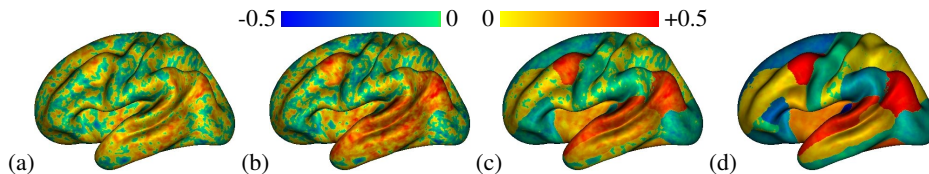

Figure 2: Normalized **w** of the left hemisphere when the SVM is regularized with a cortical atlas [31]: (a) $\beta = 0$ (no prior), (b) $\beta = 1$, (c) $\beta = 2$, (d) $\beta = 3$.

## 7 Discussion

In this contribution, we proposed to use regularization operators to add spatial consistency to SVMs for brain image analysis. We show that this provides a flexible approach to model different types of proximity between the features. We proposed derivations for both 3D image features, such as tissue maps, or surface characteristics, such as cortical thickness. We considered two different types of formulations: a discrete viewpoint in which the proximity is encoded via a graph, and a continuous viewpoint in which the data lies on a Riemannian manifold. In particular, the latter viewpoint is useful for surface cases because it overcomes problems due to surface parameterization. This paper introduced two different types of proximity. We first considered the case of regularization based on spatial proximity, which results in spatially consistent OMH making their anatomical interpretation more meaningful. We then considered a different type of proximity which allows modeling higher-level knowledge, which we call anatomical proximity. In this model, two voxels are considered close if they belong to the same brain network. For example, two voxels can be close if they belong to the same anatomical region. This can be seen as a "short-range" connectivity. Another example is that of "long-range" proximity which models the fact that distant voxels can be anatomically connected, through white matter tracts, or functionally connected, based on fMRI networks.

Preliminary evaluation was performed on 30 patients with AD and 30 age-matched controls. The results demonstrate that the proposed approaches allow obtaining spatially and anatomically coherent discrimination patterns. In particular, the obtained hyperplanes are largely consistent with the neuropathology of AD, with highly discriminant features in the medial temporal lobe, as well as lateral temporal, parietal associative and frontal areas. As for the classification results, they were comparable to those reported in the literature for AD classification (e.g. [5, 8, 7]). The use of regularization did not substantially improve the accuracy. However, the most important point is that the proposed approach makes the results more consistent with the anatomy, making their interpretation more meaningful.

Finally, it should be noted that the proposed approach is not specific to structural MRI, and can be applied to other pathologies and other types of data (e.g. functional or diffusion-weighted MRI).

### Acknowledgments

This work was supported by ANR (project HM-TC, number ANR-09-EMER-006).
Data collection and sharing for this project was funded by the Alzheimer's Disease Neuroimaging Initiative (ADNI; Principal Investigator: Michael Weiner; NIH grant U01 AG024904). ADNI data are disseminated by the Laboratory of Neuro Imaging at the University of California, Los Angeles.

## Footnotes

[1] www.loni.ucla.edu/ADNI

# References

[1] J. Ashburner and K.J. Friston. Voxel-based morphometry–the methods. *NeuroImage*, 11(6):805–21, 2000.

[2] C. Davatzikos. Why voxel-based morphometric analysis should be used with great caution when characterizing group differences. *NeuroImage*, 23(1):17–20, 2004.

[3] V.N. Vapnik. *The Nature of Statistical Learning Theory*. Springer-Verlag, 1995.

[4] B. Schölkopf and A.J. Smola. *Learning with Kernels*. MIT Press, 2001.

[5] Z. Lao et al. Morphological classification of brains via high-dimensional shape transformations and machine learning methods. *NeuroImage*, 21(1):46–57, 2004.

[6] Y. Fan et al. COMPARE: classification of morphological patterns using adaptive regional elements. *IEEE TMI*, 26(1):93–105, 2007.

[7] S. Klöppel et al. Automatic classification of MR scans in Alzheimer's disease. *Brain*, 131(3):681–9, 2008.

[8] P. Vemuri et al. Alzheimer's disease diagnosis in individual subjects using structural MR images: validation studies. *NeuroImage*, 39(3):1186–97, 2008.

[9] J. Ashburner et al. A fast diffeomorphic image registration algorithm. *NeuroImage*, 38(1):95–113, 2007.

[10] O. Querbes et al. Early diagnosis of Alzheimer's disease using cortical thickness: impact of cognitive reserve. *Brain*, 132(8):2036, 2009.

[11] J. Shawe-Taylor and N. Cristianini. *Kernel methods for pattern analysis*. Cambridge Univ Pr, 2004.

[12] D. Decoste and B. Schölkopf. Training invariant support vector machines. *Machine Learning*, 46(1):161–90, 2002.

[13] B. Schölkopf et al. Incorporating invariances in support vector learning machines. In *Proc. ICANN 1996*, page 47. Springer Verlag, 1996.

[14] B. Schölkopf et al. Prior knowledge in support vector kernels. In *Proc. conference on Advances in neural information processing systems'97*, pages 640–46. MIT Press, 1998.

[15] A.J. Smola and B. Schölkopf. On a kernel-based method for pattern recognition, regression, approximation, and operator inversion. *Algorithmica*, 22(1/2):211–31, 1998.

[16] F.R.K. Chung. *Spectral Graph Theory*. Number 92. AMS, 1992.

[17] R. I. Kondor and J.D. Lafferty. Diffusion kernels on graphs and other discrete input spaces. In *Proc. International Conference on Machine Learning*, pages 315–22, 2002.

[18] L. Gómez-Chova et al. Semi-supervised image classification with Laplacian support vector machines. *IEEE Geo Rem Sens Let*, 5(3):336–40, 2008.

[19] A.J. Smola and R. Kondor. Kernels and regularization on graphs. In *Proc. COLT*, page 144. Springer Verlag, 2003.

[20] F. Rapaport et al. Classification of microarray data using gene networks. *BMC bioinformatics*, 8(1):35, 2007.

[21] J. Jost. *Riemannian geometry and geometric analysis*. Springer Verlag, 2008.

[22] J. Lafferty and G. Lebanon. Diffusion kernels on statistical manifolds. *JMLR*, 6:129–63, 2005.

[23] A. Andrade et al. Detection of fMRI activation using cortical surface mapping. *Hum Brain Mapp*, 12(2):79–93, 2001.

[24] A. Cachia et al. A primal sketch of the cortex mean curvature: a morphogenesis based approach to study the variability of the folding patterns. *IEEE TMI*, 22(6):754–765, 2003.

[25] M.K. Chung. Heat kernel smoothing and its application to cortical manifolds. Technical report, 1090. Department of Statistics, Univ of Wisconsin, Madison, 2004.

[26] M.K. Chung et al. Cortical thickness analysis in autism with heat kernel smoothing. *NeuroImage*, 25(4):1256–65, 2005.

[27] S.-I. Amari et al. *Differential Geometry in Statistical Inference*, volume 10. Institute of Mathematical Statistics, 1987.

[28] O. Druet et al. *Blow-up theory for elliptic PDEs in Riemannian geometry*. Princeton Univ Pr, 2004.

[29] C.R.Jr Jack et al. The Alzheimer's Disease Neuroimaging Initiative (ADNI): MRI methods. *J Magn Reson Imaging*, 27(4):685–91, 2008.

[30] J. Ashburner and K.J. Friston. Unified segmentation. *NeuroImage*, 26(3):839–51, 2005.

[31] R. S. Desikan et al. An automated labeling system for subdividing the human cerebral cortex on MRI scans into gyral based regions of interest. *Neuroimage*, 31(3):968–980, 2006.

